# Cascaded Classification Models:
# Combining Models for Holistic Scene Understanding

**Geremy Heitz**       **Stephen Gould**
Department of Electrical Engineering
Stanford University, Stanford, CA 94305
{gaheitz,sgould}@stanford.edu

**Ashutosh Saxena**       **Daphne Koller**
Department of Computer Science
Stanford University, Stanford, CA 94305
{asaxena,koller}@cs.stanford.edu

## Abstract

One of the original goals of computer vision was to fully understand a natural scene. This requires solving several sub-problems simultaneously, including object detection, region labeling, and geometric reasoning. The last few decades have seen great progress in tackling each of these problems in isolation. Only recently have researchers returned to the difficult task of considering them jointly. In this work, we consider learning a set of related models in such that they both solve their own problem and help each other. We develop a framework called Cascaded Classification Models (**CCM**), where repeated instantiations of these classifiers are coupled by their input/output variables in a cascade that improves performance at each level. Our method requires only a limited "black box" interface with the models, allowing us to use very sophisticated, state-of-the-art classifiers without having to look under the hood. We demonstrate the effectiveness of our method on a large set of natural images by combining the subtasks of scene categorization, object detection, multiclass image segmentation, and 3d reconstruction.

## 1   Introduction

The problem of "holistic scene understanding" encompasses a number of notoriously difficult computer vision tasks. Presented with an image, scene understanding involves processing the image to answer a number of questions, including: (i) What type of scene is it (e.g., urban, rural, indoor)? (ii) What meaningful regions compose the image? (iii) What objects are in the image? (iv) What is the 3d structure of the scene? (See Figure 1). Many of these questions are coupled—e.g., a car present in the image indicates that the scene is likely to be urban, which in turn makes it more likely to find road or building regions. Indeed, this idea of communicating information between tasks is not new and dates back to some of the earliest work in computer vision (e.g., [1]). In this paper, we present a framework that exploits such dependencies to answer questions about novel images.

While our focus will be on image understanding, the goal of combining related classifiers is relevant to many other machine learning domains where several related tasks operate on the same (or related) raw data and provide correlated outputs. In the area of natural language processing, for instance, we might want to process a single document and predict the part of speech of all words, correspond the named entities, and label the semantic roles of verbs. In the area of audio signal processing, we might want to simultaneously do speech recognition, source separation, and speaker recognition.

In the problem of scene understanding (as in many others), state-of-the-art models already exist for many of the tasks of interest. However, these carefully engineered models are often tricky to modify, or even simply to re-implement from available descriptions. As a result, it is sometimes desirable to treat these models as "black boxes," where we have we have access only to a very simple input/output interface. in short, we require only the ability to train on data and produce classifications for each data instance; specifics are given in Section 3 below.

In this paper, we present the framework of Cascaded Classification Models (**CCM**s), where state-of-the-art "black box" classifiers for a set of related tasks are combined to improve performance on

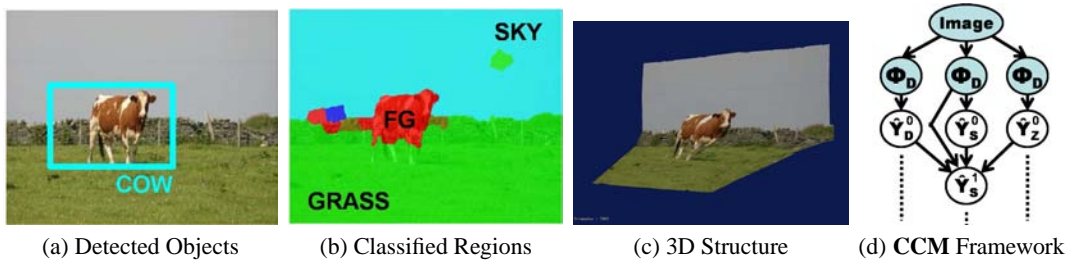

| (a) Detected Objects | (b) Classified Regions | (c) 3D Structure | (d) **CCM** Framework |

**Figure 1:** (a)-(c) Some properties of a scene required for holistic scene understanding that we seek to unify using a cascade of classifiers. (d) The **CCM** framework for jointly predicting each of these label types.

some or all tasks. Specifically, the **CCM** framework creates multiple instantiations of each classifier, and organizes them into tiers where models in the first tier learn in isolation, processing the data to produce the best classifications given only the raw instance features. Lower tiers accept as input both the features from the data instance, as well as features computed from the output classifications of the models at the previous tier. While only demonstrated in the computer vision domain, we expect the **CCM** framework have broad applicability to many applications in machine learning.

We apply our model to the scene understanding task by combining scene categorization, object detection, multi-class segmentation, and 3d reconstruction. We show how "black-box" classifiers can be easily integrated into our framework. Importantly, in extensive experiments on large image databases, we show that our combined model yields superior results on *all* tasks considered.

## 2 Related Work

A number of works in various fields aim to combine classifiers to improve final output accuracy. These works can be divided into two broad groups. The first is the combination of classifiers that predict the *same* set of random variables. Here the aim is to improved classifications by combining the outputs of the individual models. Boosting [6], in which many weak learners are combined into a highly accurate classifier, is one of the most common and powerful such scemes. In computer vision, this idea has been very successfully applied to the task of face detection using the so-called Cascade of Boosted Ensembles (CoBE) [18, 2] framework. While similar to our work in constructing a cascade of classifiers, their motivation was computational efficiency, rather than a consideration of contextual benefits. Tu [17] learns context cues by cascading models for pixel-level labeling. However, the context is, again, limited to interactions between labels of the same type.

The other broad group of works that combine classifiers is aimed at using the classifiers as components in large intelligent systems. Kumar and Hebert [9], for example, develop a large MRF-based probabilistic model linking multiclass segmentation and object detection. Such approaches have also been used in the natural language processing literature. For example, the work of Sutton and McCallum [15] combines a parsing model with a semantic role labeling model into a unified probabilistic framework that solves both simultaneously. While technically-correct probabilistic representations are appealing, it is often painful to fit existing methods into a large, complex, highly interdependent network. By leveraging the idea of cascades, our method provides a simplified approach that requires minimal tuning of the components.

The goal of holistic scene understanding dates back to the early days of computer vision, and is highlighted in the "intrinsic images" system proposed by Barrow and Tenenbaum [1], where maps of various image properties (depth, reflectance, color) are computed using information present in other maps. Over the last few decades, however, researchers have instead targeted isolated computer vision tasks, with considerable success in improving the state-of-the-art. For example, in our work, we build on the prior work in scene categorization of Li and Perona [10], object detection of Dalal and Triggs [4], multi-class image segmentation of Gould et al. [7], and 3d reconstruction of Saxena et al. [13]. Recently, however, researchers have returned to the question of how one can benefit from exploiting the dependencies between different classifiers.

Torralba et al. [16] use context to significantly boost object detection performance, and Sudderth et al. [14] use object recognition for 3d structure estimation. In independent contemporary work, Hoiem et al. [8] propose an innovative system for integrating the tasks of object recognition, surface orientation estimation, and occlusion boundary detection. Like ours, their system is modular and leverages state-of-the-art components. However, their work has a strong leaning towards 3d scene

reconstruction rather than understanding, and their algorithms contain many steps that have been specialized for this purpose. Their training also requires intimate knowledge of the implementation of each module, while ours is more flexible allowing integration of many related vision tasks regardless of their implementation details. Furthermore, we consider a broader class of images and object types, and label regions with specific classes, rather than generic properties.

## 3 Cascaded Classification Models

Our goal is to classify various characteristics of our data using state-of-the-art methods in a way that allows the each model to benefit from the others' expertise. We are interested in using proven "off-the-shelf" classifiers for each subtask. As such these classifiers will be treated as "black boxes," each with its own (specialized) data structures, feature sets, and inference and training algorithms.

To fit into our framework, we only require that each classifier provides a mechanism for including additional (auxiliary) features from other modules. Many state-of-the-art models lend themselves to the easy addition of new features. In the case of "intrinsic images" [1], the output of each component is converted into an image-sized feature map (e.g., each "pixel" contains the probability that it belongs to a car). These maps can easily be fed into the other components as additional image channels. In cases where this cannot be done, it is trivial to convert the original classifier's output to a log-odds ratio and use it along with features from their other classifiers in a simple logistic model.

A standard setup has, say, two models that predict the variables $\mathbf{Y}_D$ and $\mathbf{Y}_S$ respectively for the same input instance $\mathcal{I}$. For example, $\mathcal{I}$ might be an image, and $\mathbf{Y}_D$ could be the locations of all cars in the image, while $\mathbf{Y}_S$ could be a map indicating which pixels are road. Most algorithms begin by processing $\mathcal{I}$ to produce a set of features, and then learn a function that maps these features into a predicted label (and in some cases also a confidence estimate). Cascaded Classification Models (**CCM**s) is a joint classification model that shares information between tasks by linking component classifiers in order to leverage their relatedness. Formally:

**Definition 3.1:** An $L$-*tier* Cascaded Classification Model ($L$-**CCM**) is a cascade of classifiers of the target labels $\mathcal{Y} = \{\mathbf{Y}_1, \ldots, \mathbf{Y}_K\}^L$ ($L$ "copies" of each label) consisting of **independent** classifiers $f_{k,0}(\phi_k(\mathcal{I}); \theta_{k,0}) \rightarrow \hat{\mathbf{Y}}_k^0$ and a series of **conditional** classifiers $f_{k,\ell}(\phi_k(\mathcal{I}, \mathbf{y}_{-k}^{\ell-1}); \theta_{c,\ell}) \rightarrow \hat{\mathbf{Y}}_k^\ell$, indexed by $\ell$, indicating the "tier" of the model, where $\mathbf{y}_{-k}$ indicates the assignment to all labels *other than* $\mathbf{y}_k$. The labels at the final tier ($L-1$) represent the final classification outputs. ∎

A **CCM** uses $L$ copies of each component model, stacked into tiers, as depicted in Figure 1(d). One copy of each model lies in the first tier, and learns with only the image features, $\phi_k(\mathcal{I})$, as input. Subsequent tiers of models accepts a feature vector, $\phi_k(\mathcal{I}, \mathbf{y}_{-k}^{\ell-1})$, containing the original image features and additional features computed from the outputs of models in the preceeding tier. Given a novel test instance, classification is performed by predicting the most likely (MAP) assignment to each of the variables in the final tier.

We learn our **CCM** in a feed-forward manner. That is, we begin from the top level, training the independent ($f_{k,0}$) classifiers first, in order to maximize the classification performance on the training data. Because we assume a learning interface into each model, we simply supply the subset of data that has ground labels for that model to its learning function. For learning each component $k$ in each subsequent level $\ell$ of the **CCM**, we first perform classification using the $(\ell-1)$-tier **CCM** that has already been trained. From these output assignments, each classifier can compute a new set of features and perform learning using the algorithm of choice for that classifier.

For learning a **CCM**, we assume that we have a dataset of fully or partially annotated instances. It is not necessary for every instance to have groundtruth labels for every component, and our method works even when the training sets are disjoint. This is appealing since the prevalence of large, volunteer-annotated datasets (e.g., the LabelMe dataset [12] in vision or the Penn Treebank [11] in language processing), is likely to provide large amounts of heterogeneously labeled data.

## 4 CCM for Holistic Scene Understanding

Our scene understanding model uses a **CCM** to combine various subsets of four computer vision tasks: scene categorization, multi-class image segmentation, object detection, and 3d reconstruction. We first introduce the notation for the target labels and then briefly describe the specifics of each component. Consider an image $\mathcal{I}$. Our scene categorization classifier produces a scene label $C$ from one of a small number of classes. Our multi-class segmentation model produces a class label $S_j$

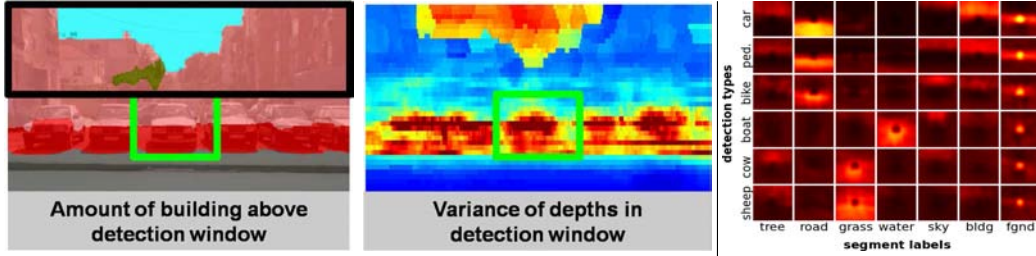

**Figure 2:** (left,middle) Two exmaple features used by the "context" aware object detector. (right) Relative location maps showing the relative location of regions (columns) to objects (rows). Each map shows the prevalence of the region relative to the center of object. For example, the top row shows that cars are likely to have road beneath and sky above, while the bottom rows show that cows and sheep are often surrounded by grass.

for each of a predefined set of regions $j$ in the image. The base object detectors produce a set of scored windows $(W_{c,i})$ that potentially contain an object of type $c$. We attach a label $D_{c,i}$ to each window, that indicates whether or not the window contains the object. Our last component module is monocular 3d reconstruction, which produces a depth $Z_i$ for every pixel $i$ in the image.

**Scene Categorization** Our scene categorization module is a simple multi-class logistic model that classifies the entire scene into one of a small number of classes. The base model uses a 13 dimensional feature vector $\phi(\mathcal{I})$ with elements based on mean and variance of RGB and YCrCb color channels over the entire image, plus a bias term. In the conditional model, we include features that indicate the relative proportions of each region label (a histogram of $S_j$ values) in the image, plus counts of the number of objects of each type detected, producing a final feature vector of length 26.

**Multiclass Image Segmentation** The segmentation module aims to assign a label to each pixel. We base our model on the work of Gould et al. [7] who make use of relative location—the preference for classes to be arranged in a consistent configuration with respect to one another (e.g., cars are often found above roads). Each image is pre-partitioned into a set $\{S_1, \ldots, S_N\}$ of regions (superpixels) and the pixels are labeled by assigning a class to each region $S_j$. The method employs a pairwise conditional Markov random field (CRF) constructed over the superpixels with node potentials based on appearance features and edge potentials encoding a preference for smoothness.

In our work we wish to model the relative location between detected objects and region labels. This has the advantage of being able to encode scale, which was not possible in [7]. The right side of Figure 2 shows the relative location maps learned by our model. These maps model the spatial location of all classes given the location and scale of detected objects. Because the detection model provides probabilities for each detection, we actually use the relative location maps multiplied by the probability that each detection is a true detection. Preliminary results showed an improvement in using these soft detections over hard (thresholded) detections.

**Object Detectors** Our detection module builds on the HOG detector of Dalal and Triggs [4]. For each class, the HOG detector is trained on a set of images disjoint from our datasets below. This detector is then applied to all images in our dataset with a low threshold that produces an overdetection. For each image $\mathcal{I}$, and each object class $c$, we typically find 10-100 candidate detection windows $W_{c,i}$. Our independent detector model learns a logistic model over a small feature vector $\phi_{c,i}$ that can be extracted directly from the candidate window.

Our conditional classifier seeks to improve the accuracy of the HOG detector by using image segmentation (denoted by $S_j$ for each region $j$), 3d reconstruction of the scene, with depths ($Z_j$) for each region, and a categorization of the scene as a whole ($C$), to improve the results of the HOG detector. Thus, the output from other modules and the image are combined into a feature vector $\phi_k(\mathcal{I}, C, \mathbf{S}, \mathbf{Z})$. A sampling of some features used are shown in Figure 2. This augmented feature vector is used in a logistic model as in the independent case. Both the independent and context aware logistics are regularized with a small ridge term to prevent overfitting.

**Reconstruction Module** Our reconstruction module is based on the work of Saxena et al. [13]. Our Markov Random Field (MRF) approach models the 3d reconstruction (i.e., depths $Z$ at each point in the image) as a function of the image features and also models the relations between depths at

|  | CAR | PEDES. | BIKE | BOAT | SHEEP | COW | **Mean** | **Segment** | **Category** |
|---|---|---|---|---|---|---|---|---|---|
| **HOG** | 0.39 | 0.29 | 0.13 | 0.11 | 0.19 | 0.28 | 0.23 | N/A | N/A |
| **Independent** | 0.55 | 0.53 | 0.57 | 0.31 | 0.39 | 0.49 | 0.47 | 72.1% | 70.6% |
| **2-CCM** | 0.58 | 0.55 | **0.65** | **0.48** | **0.45** | 0.53 | **0.54** | 75.0% | **77.3%** |
| **5-CCM** | **0.59** | **0.56** | 0.63 | 0.47 | 0.40 | **0.54** | 0.53 | **75.8%** | 76.8% |
| **Ground** | 0.49 | 0.53 | 0.62 | 0.35 | 0.40 | 0.51 | 0.48 | 73.6% | 69.9% |
| **Ideal Input** | 0.63 | 0.64 | 0.56 | 0.65 | 0.45 | 0.56 | 0.58 | 78.4% | 86.7% |

**Table 1:** Numerical evaluation of our various training regimes for the **DS1** dataset. We show average precision (AP) for the six classes, as well as the mean. We also show segmentation and scene categorization accuracy.

various points in the image. For example, unless there is occlusion, it is more likely that two nearby regions in the image would have similar depths.

More formally, our variables are continuous, i.e., at a point $i$, the depth $Z_i \in \mathbb{R}$. Our baseline model consists of two types of terms. The first terms model the depth at each point as a linear function of the local image features, and the second type models relationships between neighboring points, encouraging smoothness. Our conditional model includes an additional set of terms that models the depth at each point as a function of the features computed from an image segmentation $\mathbf{S}$ in the neighborhood of a point. By including this third term, our model benefits from the segmentation outputs in various ways. For example, a classification of grass implies a horizontal surface, and a classification of sky correlates with distant image points. While detection outputs might also help reconstruction, we found that most of the signal was present in the segmentation maps, and therefore dropped the detection features for simplicity.

## 5 Experiments

We perform experiments on two subsets of images. The first subset **DS1** contains 422 fully-labeled images of urban and rural outdoor scenes. Each image is assigned a category (*urban*, *rural*, *water*, *other*). We hand label each pixel as belonging to one of: *tree*, *road*, *grass*, *water*, *sky*, *building* and *foreground*. The foreground class captures detectable objects, and a *void* class (not used during training or evaluation) allows for the small number of regions not fitting into one of these classes (e.g., mountain) to be ignored. This is standard practice for the pixel-labeling task (e.g., see [3]). We also annotate the location of six different object categories (*car*, *pedestrian*, *motorcycle*, *boat*, *sheep*, and *cow*) by drawing a tight bounding box around each object. We use this dataset to demonstrate the combining of three vision tasks: object detection, multi-class segmentation, and scene categorization using the models described above.

Our much larger second dataset **DS2** was assembled by combining 362 images from the **DS1** dataset (including either the segmentation or detection labels, but not both), 296 images from the Microsoft Research Segmentation dataset [3] (labeled with segments), 557 images from the PASCAL VOC 2005 and 2006 challenges [5] (labeled with objects), and 534 images with ground truth depth information. This results in 1749 images with disjoint labelings (no image contains groundtruth labels for more than one task). Combining these datasets results in 534 reconstruction images with groundtruth depths obtained by laser range-finder (split into 400 training and 134 test), 596 images with groundtruth detections (same 6 classes as above, split into 297 train and 299 test), and 615 with groundtruth segmentations (300 train and 315 test). This dataset demonstrates the typical situation in learning related tasks whereby it is difficult to obtain large fully-labeled datasets. We use this dataset to demonstrate the power of our method in leveraging the data from these three tasks to improve performance.

### 5.1 DS1 Dataset

Experiments with the **DS1** dataset were performed using 5-fold cross validation, and we report the mean performance results across folds. We compare five training/testing regimes (see Table 1). **Independent** learns parameters on a 0-Tier (independent) **CCM**, where no information is exchanged between tasks. We compare two levels of complexity for our method, a **2-CCM** and a **5-CCM** to test how the depth of the cascade affects performance. The last two training/testing regimes involve using groundtruth information at every stage for training and for both training and testing, respectively. **Groundtruth** trains a 5-**CCM** using groundtruth inputs for the feature construction (i.e., as if each tier received perfect inputs from above), but is evaluated with real inputs. The **Ideal**

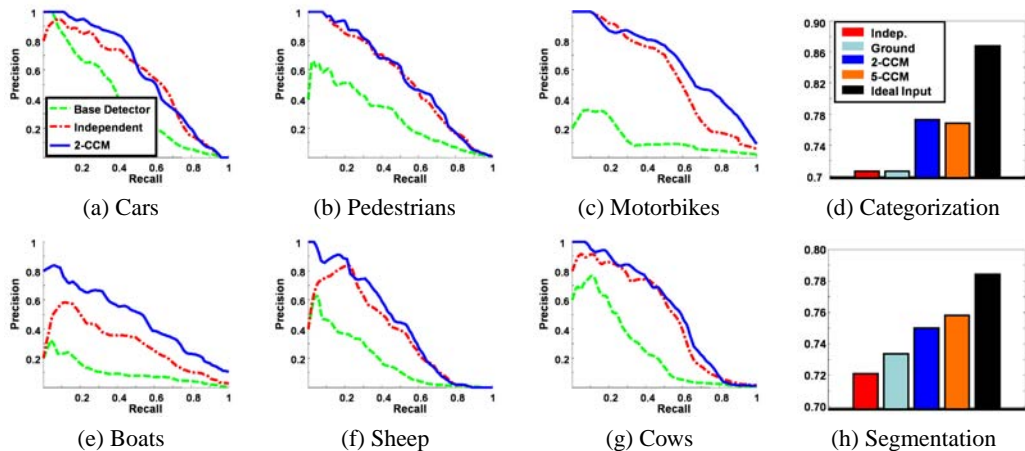

**Figure 3:** Results for the DS1 dataset. (a-c,e-g) show precision-recall curves for the six object classes that we consider, while (d) shows our accuracy on the scene categorization task and (h) our accuracy in labeling regions in one of seven classes.

**Input** experiment uses the **Groundtruth** model and also uses the groundtruth input to each tier *at testing time*. We could do this since, for this dataset, we had access to fully labeled groundtruth. Obviously this is not a legitimate operating mode, but does provide an interesting upper bound on what we might hope to achieve.

To quantitatively evaluate our method, we consider metrics appropriate to the tasks in question. For scene categorization, we report an overall accuracy for assigning the correct scene label to an image. For segmentation, we compute a per-segment accuracy, where each segment is assigned the groundtruth label that occurs for the majority of pixels in the region. For detection, we consider a particular detection correct if the overlap score is larger than 0.2 (overlap score equals the area of intersection divided by the area of union between the detected bounding box and the groundtruth). We plot precision-recall (PR) curves for detections, and report the average precision of these curves. AP is a more stable version of the area under the PR curve.

Our numerical results are shown in Table 1, and the corresponding graphs are given in Figure 3. The PR curves compare the HOG detector results to our **Independent** results and to our **2-CCM** results. It is interesting to note that a large gain was achieved by adding the independent features to the object detectors. While the HOG score looks at only the pixels inside the target window, the other features take into account the size and location of the window, allowing our model to capture the fact that foreground object tend to occur in the middle of the image and at a relatively small range of scales. On top of this, we were able to gain an additional benefit through the use of context in the **CCM** framework. For the categorization task, we gained 7% using the **CCM** framework, and for segmentation, **CCM** afforded a 3% improvement in accuracy. Furthermore, for this task, running an additional three tiers, for a 5-**CCM**, produced an additional 1% improvement.

Interestingly, the **Groundtruth** method performs little better than **Independent** for these three tasks. This shows that it is better to train the models using input features that are closer to the features it will see at test time. In this way, the downstream tiers can learn to ignore signals that the upstream tiers are bad at capturing, or even take advantage of consistent upstream bias. Also, the **Ideal Input** results show that **CCM**s have made significant progress towards the best we can hope for from these models.

## 5.2   DS2 Dataset

For this dataset we combine the three subtasks of reconstruction, segmentation, and object detection. Furthermore, as described above, the labels for our training data are disjoint. We trained an **Independent** model and a 2-**CCM** on this data. Quantitatively, 2-**CCM** outperformed **Independent** on segmentation by 2% (75% vs. 73% accuracy), on detection by 0.02 (0.33 vs. 0.31 mean average precision), and on depth reconstruction by 1.3 meters (15.4 vs. 16.7 root mean squared error).

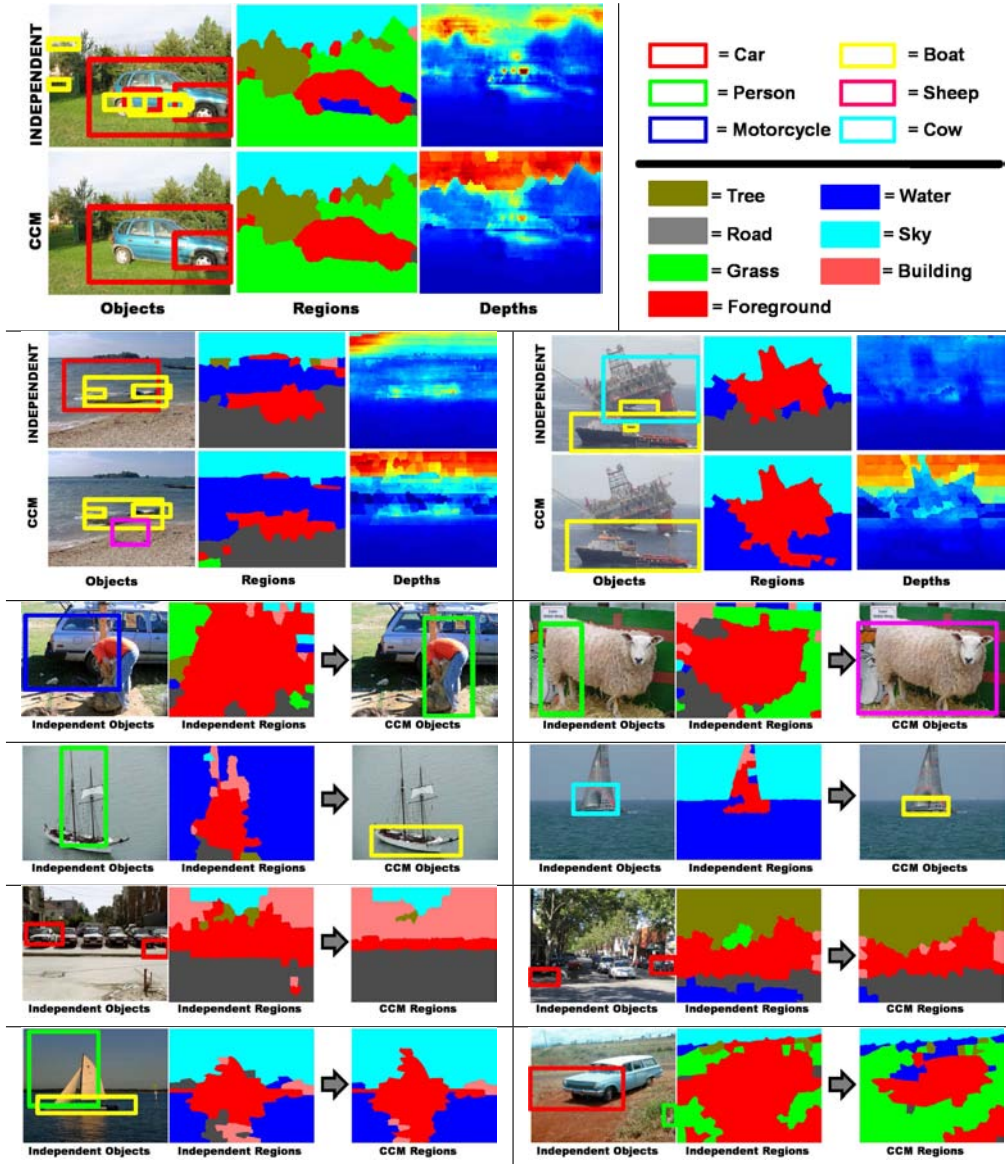

**Figure 4:** (top two rows) three cases where **CCM** improved results for all tasks. In the first, for instance, the presence of grass allows the **CCM** to remove the boat detections. The next four rows show four examples where detections are improved and four examples where segmentations are improved.

Figure 4 shows example outputs from each component. The first three (top two rows) show images where all components improved over the independent model. In the top left our detectors removed some false boat detections which were out of context and determined that the watery appearance of the bottom of the car was actually foreground. Also by providing a sky segment, our method allowed the 3d reconstruction model to infer that those pixels must be very distant (red). The next two examples show similar improvement for detections of boats and water.

The remaining examples show how separate tasks improve by using information from the others. In each example we show results from the independent model for the task in question, the independent contextual task and the 2-**CCM** output. The first four examples show that our method was able to make correct detections whereas the independent model could not. The last examples show improvements in multi-class image segmentation.

# 6 Discussion

In this paper, we have presented the Cascaded Classification Models (**CCM**) method for combining a collection of state-of-the-art classifiers toward improving the results of each. We demonstrated our method on the task of holistic scene understanding by combining scene categorization, object detection, multi-class segmentation and depth reconstruction, and improving on all. Our results are consistent with other contemporary research, including the work of Hoiem et al. [8], which uses different components and a smaller number of object classes.

Importantly, our framework is very general and can be applied to a number of machine learning domains. This result provides hope that we can improve by combining our complex models in a simple way. The simplicity of our method is one of its most appealing aspects. Cascades of classifiers have been used extensively within a particular task, and our results suggest that this should generalize to work between tasks. In addition, we showed that **CCM**s can benefit from the cascade even with disjoint training data, e.g., no images containing labels for more than one subtask.

In our experiments, we passed relatively few features between the tasks. Due to the homogeneity of our data, many of the features carried the same signal (e.g., a high probability of an ocean scene is a surrogate for a large portion of the image containing water regions). For larger, more heterogeneous datasets, including more features may improve performance. In addition, larger datasets will help prevent the overfitting that we experienced when trying to include a large number of features.

It is an open question how deep a **CCM** is appropriate in a given scenario. Overfitting is anticipated for very deep cascades. Furthermore, because of limits in the context signal, we cannot expect to get unlimited improvements. Further exploration of cases where this combination is appropriate is an important future direction. Another exciting avenue is the idea of feeding back information from the later classifiers to the earlier ones. Intuitively, a later classifier might encourage earlier ones to focus its effort on fixing certain error modes, or allow the earlier classifiers to ignore mistakes that do not hurt "downstream." This also should allow components with little training data to optimize their results to be most beneficial to other modules, while worrying less about their own task.

**Acknowledgements** This work was supported by the DARPA Transfer Learning program under contract number FA8750-05-2-0249 and the Multidisciplinary University Research Initiative (MURI), contract number N000140710747, managed by the Office of Naval Research.

# References

[1] H. G. Barrow and J.M. Tenenbaum. Recovering intrinsic scene characteristics from images. *CVS*, 1978.

[2] S.C. Brubaker, J. Wu, J. Sun, M.D. Mullin, and J.M. Rehg. On the design of cascades of boosted ensembles for face detection. In *Tech report GIT-GVU-05-28*, 2005.

[3] A. Criminisi. Microsoft research cambridge object recognition image database (version 1.0 and 2.0)., 2004. Available Online: http://research.microsoft.com/vision/cambridge/recognition.

[4] N. Dalal and B. Triggs. Histograms of oriented gradients for human detection. In *CVPR*, 2005.

[5] M. Everingham et al. The 2005 pascal visual object classes challenge. In *MLCW*, 2005.

[6] Y. Freund and R.E. Schapire. A decision-theoretic generalization of on-line learning and an application to boosting. In *European Conference on Computational Learning Theory*, pages 23–37, 1995.

[7] S. Gould, J. Rodgers, D. Cohen, G. Elidan, and D. Koller. Multi-class segmentation with relative location prior. *IJCV*, 2008.

[8] D. Hoiem, A.A. Efros, and M. Hebert. Closing the loop on scene interpretation, 2008.

[9] S. Kumar and M. Hebert. A hier. field framework for unified context-based classification. In *ICCV*, 2005.

[10] F. Li and P. Perona. A bayesian hier. model for learning natural scene categories. In *CVPR*, 2005.

[11] M. P. Marcus, M.A. Marcinkiewicz, and B. Santorini. Building a large annotated corpus of english: the penn treebank. *Comput. Linguist.*, 19(2), 1993.

[12] B.C. Russell, A.B. Torralba, K.P. Murphy, and W.T. Freeman. Labelme: A database and web-based tool for image annotation. *IJCV*, 2008.

[13] A. Saxena, M. Sun, and A.Y. Ng. Learning 3-d scene structure from a single still image. In *PAMI*, 2008.

[14] E.B. Sudderth, A. Torralba, W.T. Freeman, and A.S. Willsky. Depth from familiar objects: A hierarchical model for 3d scenes. In *CVPR*, 2006.

[15] C. Sutton and A. McCallum. Joint parsing and semantic role labeling. In *CoNLL*, 2005.

[16] Antonio B. Torralba, Kevin P. Murphy, and William T. Freeman. Contextual models for object detection using boosted random fields. In *NIPS*, 2004.

[17] Z. Tu. Auto-context and its application to high-level vision tasks. In *CVPR*, 2008.

[18] P. Viola and M.J. Jones. Robust real-time object detection. *IJCV*, 2001.

